# Examples of learning curves from a modified VC-formalism.

**A. Kowalczyk & J. Szymański**
Telstra Research Laboratories
770 Blackburn Road,
Clayton, Vic. 3168, Australia
{a.kowalczyk,j.szymanski}@trl.oz.au)

**P.L. Bartlett & R.C. Williamson**
Department of Systems Engineering
Australian National University
Canberra, ACT 0200, Australia
{bartlett,williams}@syseng.anu.edu.au

## Abstract

We examine the issue of evaluation of model specific parameters in a modified VC-formalism. Two examples are analyzed: the 2-dimensional homogeneous perceptron and the 1-dimensional higher order neuron. Both models are solved theoretically, and their learning curves are compared against true learning curves. It is shown that the formalism has the potential to generate a variety of learning curves, including ones displaying "phase transitions."

## 1 Introduction

One of the main criticisms of the Vapnik-Chervonenkis theory of learning [15] is that the results of the theory appear very loose when compared with empirical data. In contrast, theory based on statistical physics ideas [1] provides tighter numerical results as well as qualitatively distinct predictions (such as "phase transitions" to perfect generalization). (See [5, 14] for a fuller discussion.) A question arises as to whether the VC-theory can be modified to give these improvements. The general direction of such a modification is obvious: one needs to sacrifice the universality of the VC-bounds and introduce model (e.g. distribution) dependent parameters. This obviously can be done in a variety of ways. Some specific examples are VC-entropy [15], empirical VC-dimensions [16], efficient complexity [17] or $(\mu, C)$-uniformity [8, 9] in a VC-formalism with error shells. An extension of the last formalism is of central interest to this paper. It is based on a refinement of the "fundamental theorem of computational learning" [2] and its main innovation is to split the set of partitions of a training sample into separate "error shells", each composed of error vectors corresponding to the different error values.

Such a split introduces a whole range of new parameters (the average number of patterns in each of a series of error shells) in addition to the VC dimension. The difficulty of determining these parameters then arises. There are some crude, "obvious" upper bounds

on them which lead to both the VC-based estimates [2, 3, 15] and the statistical physics based formalism (with phase transitions) [5] as specific cases of this novel theory. Thus there is an obvious potential for improvement of the theory with tighter bounds. In particular we find that the introduction of a single parameter (order of uniformity), which in a sense determines shifts in relative sizes of error shells, leads to a full family of shapes of learning curves continuously ranging in behavior from decay proportional to the inverse of the training sample size to "phase transitions" (sudden drops) to perfect generalization in small training sample sizes. We present initial comparison of the learning curves from this new formalism with "true" learning curves for two simple neral networks.

## 2 Overview of the formalism

The presentation is set in the typical PAC-style; the notation follows [2]. We consider a space $X$ of samples with a probability measure $\mu$, a subspace $H$ of binary functions $X \to \{0, 1\}$ (dichotomies) (called the *hypothesis space*) and a *target hypothesis* $t \in H$. For each $h \in H$ and each $m$-sample $\vec{x} = (x_1, ..., x_m) \in X^m$ ($m \in \{1, 2, ...\}$), we denote by $\epsilon_{h,\vec{x}} \stackrel{def}{=} \frac{1}{m} \sum_{i=1}^{m} |t-h|(x_i)$ *the empirical error* of $h$ on $\vec{x}$, and by $\epsilon_h \stackrel{def}{=} \int_X |t-h|(x)\mu(dx)$ the expected error of $h \in H$.

For each $m \in \{1, 2, ...\}$ let us consider the random variable

$$\epsilon_H^{max}(\vec{x}) \stackrel{def}{=} \max_{h \in H}\{\epsilon_h \; ; \; \epsilon_{h,\vec{x}} = 0\} \qquad (\vec{x} \in X^m) \tag{1}$$

defined as the maximal expected error of an hypothesis $h \in H$ consistent with $t$ on $\vec{x}$. The *learning curve of* $H$, defined as the expected value of $\epsilon_H^{max}$,

$$\epsilon_H^{av}(m) \stackrel{def}{=} E_{X^m}[\epsilon_H^{max}] = \int_{X^m} \epsilon_H^{max}(\vec{x})\mu^m(d\vec{x}) \qquad (\vec{x} \in X^m) \tag{2}$$

is of central interest to us. Upper bounds on it can be derived from basic PAC-estimates as follows. For $\epsilon \geq 0$ we denote by $H_\epsilon \stackrel{def}{=} \{h \in H \; ; \; \epsilon_h \geq \epsilon\}$ the subset of $\epsilon$-*bad hypotheses* and by

$$Q_\epsilon^m \stackrel{def}{=} \{\vec{x} \in X^m \; ; \; \exists_{h \in H_\epsilon} \epsilon_{h,\vec{x}} = 0\} = \{\vec{x} \in X^m \; ; \; \exists_{h \in H} \epsilon_{h,\vec{x}} = 0 \;\&\; \epsilon_h \geq \epsilon\} \tag{3}$$

the subset of $m$-samples for which there exists an $\epsilon$-bad hypothesis consistent with the target $t$.

**Lemma 1** *If* $\mu^m(Q_\epsilon^m) \leq \psi(\epsilon, m)$, *then* $\epsilon_H^{av}(m) \leq \int_0^1 \min(1, \psi(\epsilon, m))\mu(d\epsilon)$, *and equality in the assumption implies equality in the conclusion.* $\Box$

**Proof outline.** If the assumption holds, then $\Psi(\epsilon, m) \stackrel{def}{=} 1 - \min(1, \psi(\epsilon, m))$ is a lower bound on the cumulative distribution of the random variable (1). Thus $E_{X^m}[\epsilon_H^{max}] \leq \int_0^1 \epsilon \frac{d}{d\epsilon} \Psi(\epsilon, m) d\epsilon$ and integration by parts yields the conclusion.
$\Box$

Given $\vec{x} = (x_1, ..., x_m) \in X^m$, let us introduce the transformation (projection) $\pi_{t,\vec{x}} : H \to \{0, 1\}^m$ allocating to each $h \in H$ the vector

$$\pi_{t,\vec{x}}(h) \stackrel{def}{=} (|h(x_1) - t(x_1)|, ..., |h(x_m) - t(x_m)|)$$

called *the error pattern* of $h$ on $\vec{x}$. For a subset $G \subset H$, let $\pi_{t,\vec{x}}(G) = \{\pi_{t,\vec{x}}(h) : h \in G\}$. The space $\{0, 1\}^m$ is the disjoint union of *error shells* $\mathcal{E}_i^m \stackrel{def}{=} \{(\xi_1, ..., \xi_m) \in \{0, 1\}^m \; ; \; \xi_1 + \cdots + \xi_m = i\}$ for $i = 0, 1, ..., m$, and $|\pi_{t,\vec{x}}(H_\epsilon) \cap \mathcal{E}_i^m|$ is the number

of different error patterns with $i$ errors which can be obtained for $h \in H_\epsilon$. We shall employ the following notation for its average:

$$|H_\epsilon|_i^m \stackrel{def}{=} E_{X^m}[|\pi_{t,\vec{x}}(H_\epsilon) \cap \mathcal{E}_i^m|] = \int_{X^m} |\pi_{t,\vec{x}}(H_\epsilon) \cap \mathcal{E}_i^m| \mu^m(d\vec{x}). \qquad (4)$$

The central result of this paper, which gives a bound on the probability of the set $Q_\epsilon^m$ as in Lemma 1 in terms of $|H_\epsilon|_i^m$, will be given now. It is obtained by modification of the proof of [8, Theorem 1] which is a refinement of the proof of the "fundamental theorem of computational learning" in [2]. It is a simplified version (to the consistent learning case) of the basic estimate discussed in [9, 7].

**Theorem 2** *For any integer $k \geq 0$ and $0 \leq \epsilon, \gamma \leq 1$*

$$\mu^m(Q_\epsilon^m) \leq A_{\epsilon,k,\gamma} \sum_{j \geq \gamma k}^{k} \binom{k}{j} \binom{m+k}{j}^{-1} |H_\epsilon|_j^{m+k}, \qquad (5)$$

*where $A_{\epsilon,k,\gamma} \stackrel{def}{=} \left(1 - \sum_{j=0}^{\lfloor \gamma k \rfloor} \binom{k}{j} \epsilon^j (1-\epsilon)^{k-j}\right)^{-1}$, for $k > 0$ and $A_{\epsilon,0,\gamma} \stackrel{def}{=} 1$.* □

Since error shells are disjoint we have the following relation:

$$\bar{P}_H(m) \stackrel{def}{=} 2^{-m} \int_{X^m} |\pi_{\vec{x}}(H)| \mu^m(d\vec{x}) = 2^{-m} \sum_{i=0}^{m} |H|_i^m \leq \Pi_H(m)/2^m \qquad (6)$$

where $\pi_{\vec{x}}(h) \stackrel{def}{=} \pi_{0,\vec{x}}(h)$, $|H|_i^m \stackrel{def}{=} |H_0|_i^m$ and $\Pi_H(m) \stackrel{def}{=} \max_{\vec{x} \in X^m} |\pi_{\vec{x}}(H)|$ is the *growth function* [2] of $H$. (Note that assuming that the target $t \equiv 0$ does not affect the cardinality of $\pi_{t,\vec{x}}(H)$.) If the VC-dimension of $H$, $d = d_{VC}(H)$, is finite, we have the well-known estimate [2]

$$\Pi_H(m) \leq \Phi(d,m) \stackrel{def}{=} \sum_{j=0}^{d} \binom{m}{j} \leq (em/d)^d. \qquad (7)$$

**Corollary 3** *(i) If the VC-dimension $d$ of $H$ is finite and $m > 8/\epsilon$, then $\mu^m(Q_\epsilon^m) \leq 2^{2-m\epsilon/2}(2em/d)^d$.*

*(ii) If $H$ has finite cardinality, then $\mu^m(Q_\epsilon^m) \leq \sum_{h \in H_\epsilon}(1-\epsilon_h)^m$.*

**Proof.** (i) Use the estimate $A_{\epsilon,k,\epsilon/2} \leq 2$ for $k \geq 8/\epsilon$ resulting from the Chernoff bound and set $\gamma = \epsilon/2$ and $k = m$ in (5). (ii) Substitute the following crude estimate:

$$|H_\epsilon|_i^m \leq \sum_{i=0}^{m} |H_\epsilon|_i^m \leq \sum_{i=0}^{m} |H|_i^m \leq P_H \leq (em/d)^d,$$

into the previous estimate. (iii) Set $k = 0$ into (i) and use the estimate

$$|H|_i^m \leq \sum_{h \in H_\epsilon} Pr_{X^m}(\epsilon_{h,\vec{x}} = i/m) = \sum_{h \in H_\epsilon}(1-\epsilon_h)^{m-i}\epsilon_h^i. \quad □$$

The inequality in Corollary 3.i (ignoring the factor of 2) is the basic estimate of the VC-formalism (c.f. [2]); the inequality in Corollary 3.ii is the union bound which is the starting point for the statistical physics based formalism developed in [5]. In this sense both of these theories are unified in estimate (5) and all their conclusions (including the prediction

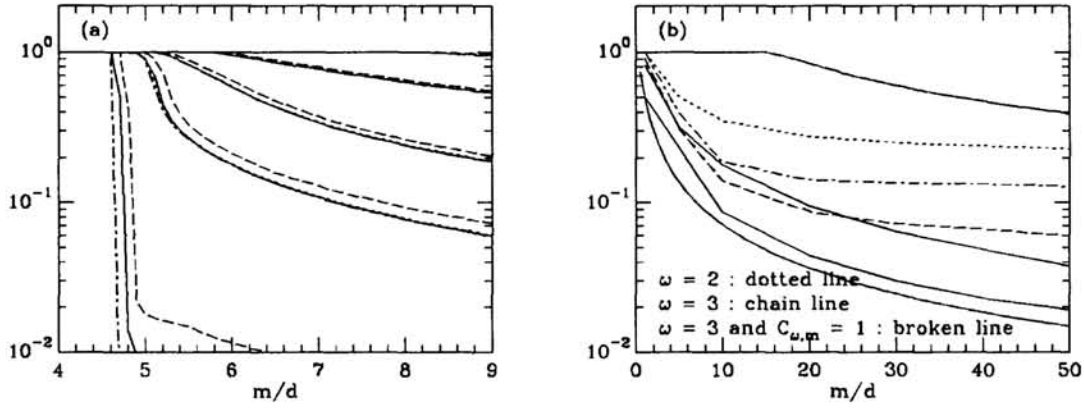

Figure 1: **(a)** Examples of upper bounds on the learning curves for the case of finite VC-dimension $d = d_{VC}(H)$ implied by Corollary 4.ii for $C_{\omega,m} \equiv$ const. They split into five distinct "bands" of four curves each, according to the values of the order of uniformity $\omega = 2, 3, 4, 5, 10$ (in the top-down order). Each band contains a solid line ($C_{\omega,m} \equiv 1, d = 100$), a dotted line ($C_{\omega,m} \equiv 100, d = 100$), a chain line ($C_{\omega,m} \equiv 1, d = 1000$) and a broken line ($C_{\omega,m} \equiv 100, d = 1000$).
**(b)** Various learning curves for the 2-dimensional homogeneous perceptron. Solid lines (top to bottom): *(i)* - for the VC-theory bound (Corollary 3.ii) with VC-dimension $d = 2$; *(ii)* - for the bound (for Eqn. 5 and Lemma 1) with $\gamma = \epsilon$, $k = m$ and the upper bounds $|H_\epsilon|_i^m \leq |H|_i^m = 2$ for $i = 1, ..., m-1$ and $|H_\epsilon|_i^m \leq |H|_i^m = 1$ for $i = 0, m$; *(iii)* - as in *(ii)* but with the exact values for $|H_\epsilon|_i^m$ as in (11); *(iv)* - true learning curve (Eqn. 13). The $\omega$-uniformity bound for $\omega = 2$ (with the minimal $C_{\omega,m}$ satisfying (9), which turn out to be $=$ const $= 1$) is shown by dotted line; for $\omega = 3$ the chain line gives the result for minimal $C_{\omega,m}$ and the broken line for $C_{\omega,m}$ set to 1.

of phase transitions to perfect generalization for the Ising perceptron for $\alpha = m/d < 1.448$ in the thermodynamic limit [5]) can be derived from this estimate, and possibly improved with the use of tighter estimates on $|H_\epsilon|_i^m$.

We now formally introduce a family of estimates on $|H_\epsilon|_i^m$ in order to discuss a potential of our formalism. For any $m$, $\epsilon$ and $\omega \geq 1.0$ there exists $C_{\omega,m} > 0$ such that

$$|H_\epsilon|_i^m \leq |H|_i^m \leq C_{\omega,m} \binom{m}{i} \bar{P}_H(m)^{1-|1-2i/m|^\omega} \qquad \text{(for } 0 \leq i \leq m\text{)}. \tag{8}$$

We shall call such an estimate *an $\omega$-uniformity bound*.

**Corollary 4** *(i) If an $\omega$-uniformity bound (8) holds, then*

$$\mu^m(Q_\epsilon^m) \leq A_{\epsilon,m,\gamma} C_{\omega,m} \sum_{j \geq \gamma m}^m \binom{m}{j} \bar{P}_H(2m)^{1-|1-j/m|^\omega}; \tag{9}$$

*(ii) if additionally $d = d_{VC}(H) < \infty$, then*

$$\mu^m(Q_\epsilon^m) \leq A_{\epsilon,m,\gamma} C_{\omega,m} \sum_{j \geq \gamma m}^m \binom{m}{j} \left(2^{-2m}(2em/d)^d\right)^{1-|1-j/m|^\omega}. \; \Box \tag{10}$$

## 3   Examples of learning curves

In this section we evaluate the above formalism on two examples of simple neural networks.

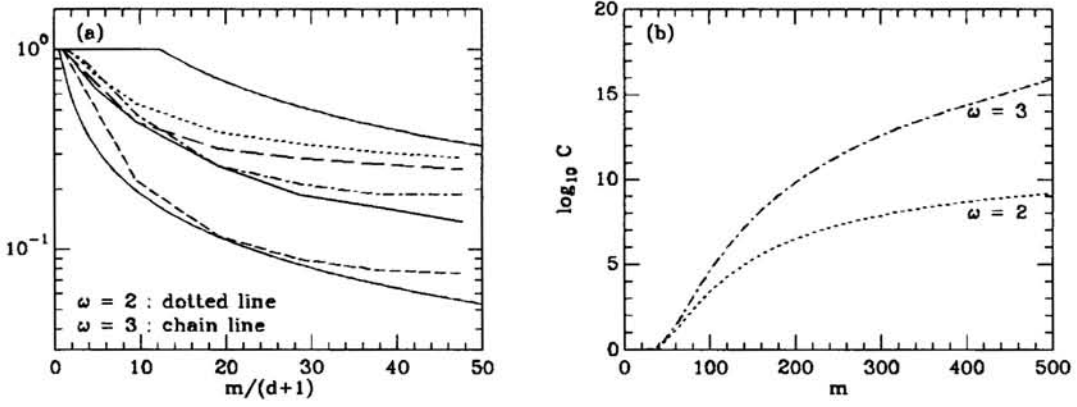

Figure 2: **(a)** Different learning curves for the higher order neuron (analogous to Fig. 1.b). Solid lines (top to bottom) *(i)* - for the VC-theory bound (Corollary 3.ii) with VC-dimension $d+1 = 21$; *(ii)* - for the bound (5) with $\gamma = \epsilon$ and the upper bounds $|H_\epsilon|_i^m \leq |H|_i^m$ with $|H|_i^m$ given by (15); *(iii)* - true learning curve (the upper bound given by (18)). The $\omega$-uniformity bound/approximation are plotted as chain and dotted lines for the minimal $C_{\omega,m}$ satisfying (8), and as broken (long broken) line for $C_{\omega,m} = const = 1$ with $\omega = 2$ ($\omega = 3$). **(b)** Plots of the minimal value of $C_{\omega,m}$ satisfying condition of $\omega$-uniformity bound (8) for higher order neuron and selected values of $\omega$.

### 3.1   2-dimensional homogeneous perceptron

We consider $X \stackrel{def}{=} \mathbf{R}^2$ and $H$ defined as the family of all functions $(\xi_1, \xi_2) \mapsto \theta(\xi_1 w_1 + \xi_2 w_2)$, where $(w_1, w_2) \in \mathbf{R}^2$ and $\theta(\tau)$ is defined as 1 if $\tau \geq 0$ and 0, otherwise, and the probability measure $\mu$ on $\mathbf{R}^2$ has rotational symmetry with respect to the origin. Fix an arbitrary target $t \in H$. In such a case

$$|H_\epsilon|_i^m = \begin{cases} 2(1-\epsilon)^m - (1-2\epsilon)^m & \text{(for } i = 0 \text{ and } 0 \leq \epsilon \leq 1/2), \\ 1 & \text{(for } i = m), \\ 2\sum_{j=0}^i \binom{m}{j} \epsilon^j (1-\epsilon)^{m-j} & \text{(otherwise).} \end{cases} \tag{11}$$

In particular we find that $|H|_i^m = 1$ for $i = 0, m$ and $|H|_i^m = 2$, otherwise, and

$$\bar{P}_H(m) = \sum_{i=0}^m |H|_i^m / 2^m = (1 + 2 + \cdots + 2 + 1)/2^m = m/2^{m-1}. \tag{12}$$

and the true learning curve is

$$\epsilon_H^{av}(m) = 1.5(m+1)^{-1}. \tag{13}$$

The latter expression results from Lemma 1 and the equality

$$\mu^m(Q_\epsilon^m) = \begin{cases} 2(1-\epsilon)^m - (1-2\epsilon)^m & \text{(for } 0 \leq \epsilon \leq 1/2), \\ 2(1-\epsilon)^m & \text{(for } 1/2 < \epsilon \leq 1). \end{cases} \tag{14}$$

Different learning curves (bounds and approximations) for homogeneous perceptron are plotted in Figure 1.b.

### 3.2   1-dimensional higher order neuron

We consider $X \stackrel{def}{=} [0, 1] \subset \mathbf{R}$ with a continuous probability distribution $\mu$. Define the hypothesis space $H \subset \{0, 1\}^X$ as the set of all functions of the form $\theta \circ p(x)$ where $p$ is a

polynomial of degree $\leq d$ on $\mathbf{R}$. Let the target be constant, $t \equiv 1$. It is easy to see that $H$ restricted to a finite subset of $[0, 1]$ is exactly the restriction of the family of all functions $\tilde{H} \subset \{0, 1\}^{[0,1]}$ with up to $d$ "jumps" from 0 to 1 or 1 to 0 and thus $d_{VC}(H) = d+1$. With probability 1 an $m$-sample $\vec{x} = (x_1, ..., x_m)$ from $X^m$ is such that $x_i \neq x_j$ for $i \neq j$. For such *a generic* $\vec{x}$, $|\pi_{t,\vec{x}}(H) \cap \mathcal{E}_i^m| = const = |H|_i^m$. This observation was used to derive the following relations for the computation of $|H|_i^m$:

$$|H|_i^m = \sum_{\delta=0}^{\min(d,m-1)} |\tilde{H}^{(\delta)}|_i^m + |\tilde{H}^{(\delta)}|_{m-i}^m, \tag{15}$$

for $0 \leq i \leq m$, where $|\tilde{H}^{(\delta)}|_i^m$, for $\delta = 0, 1, ..., d$, is defined as follows. We initialize $|\tilde{H}^{(0)}|_0^m = |\tilde{H}^{(1)}|_i^m \overset{def}{=} 1$ for $i = 1, ..., m-1, |\tilde{H}^{(1)}|_0^m = |\tilde{H}^{(1)}|_m^m \overset{def}{=} 0$ and $|\tilde{H}^{(\delta)}|_i^m \overset{def}{=} 0$ for $i = 0, 1, ..., m, \delta = 2, 3, ..., d$, and then, recurrently, for $\delta \geq 2$ we set $|\tilde{H}^{(\delta)}|_i^m \overset{def}{=}$ $\sum_{k=\max(\delta,m-i)}^{m-1} |\tilde{H}^{(\delta-1)}|_{i-m+k}^k$ if $\delta$ is odd and $|\tilde{H}^{(\delta)}|_i^m \overset{def}{=} \sum_{k=\delta}^{m-1} |\tilde{H}^{(\delta-1)}|_i^k$ if $\delta$ is even. (Here $|\tilde{H}^{(\delta)}|_i^m$ is defined by the relation (4) with the target $t \equiv 1$ for the hypothesis space $H^{(\delta)} \subset \tilde{H}$ composed of functions having the value 1 near 0 and exactly $\delta$ jumps in $(0, 1)$, exactly at entries of $\vec{x}$; similarly as for $H$, $|H^{(\delta)}|_i^m = |\pi_{1,\vec{x}} H^{(\delta)} \cap \mathcal{E}_i^m|$ for a generic $m$-sample $\vec{x} \in (0, 1)^m$.)

Analyzing an embedding of $\mathbf{R}$ into $\mathbf{R}^d$, and using an argument based on the Vandermonde determinant as in [6, 13], it can be proved that the partition function $\Pi_H$ is given by Cover's counting function [4], and that

$$\bar{P}_H(m) = \sum_{i=0}^{m} |H|_i^m / 2^m = \Pi_H(m)/2^m = 2 \sum_{i=0}^{d} \binom{m-1}{i} / 2^m. \tag{16}$$

For the uniform distribution on $[0, 1]$ and a generic $\vec{x} \in [0, 1]^m$ let $\lambda_k(\vec{x})$ denote the sum of $k$ largest segments of the partition of $[0, 1]$ into $m+1$ segments by the entries of $\vec{x}$. Then

$$\lambda_{\lfloor d/2 \rfloor}(\vec{x}) \leq \epsilon_H^{max}(\vec{x}) \leq \lambda_{\lfloor d/2 \rfloor+1}(\vec{x}). \tag{17}$$

An explicit expression for the expected value of $\lambda_k$ is known [11], thus a very tight bound on the true learning curve $\epsilon_H^{av}(m)$ defined by (2) can be obtained:

$$\frac{\lfloor d/2 \rfloor}{m+1} \left(1 + \sum_{j=\lfloor d/2 \rfloor+1}^{m+1} \frac{1}{j}\right) \leq \epsilon_H^{av}(m) \leq \frac{\lfloor d/2 \rfloor + 1}{m+1} \left(1 + \sum_{j=\lfloor d/2 \rfloor+2}^{m+1} \frac{1}{j}\right). \tag{18}$$

Numerical results are shown in Figure 2.

## 4    Discussion and conclusions

The basic estimate (5) of Theorem 1 has been used to produce upper bounds on the learning curve (via Lemma 1) in three different ways: $(i)$ using the exact values of coefficients $|H_\epsilon|_i^m$ (Fig. 1a), $(ii)$ using the estimate $|H_\epsilon|_i^m \leq |H|_i^m$ and the values of $|H|_i^m$ and $(iii)$ using the $\omega$-uniformity bound (8) with minimal value of $C_{\omega,m}$ and as an "approximation" with $C_{\omega,m} = const = 1$. Both examples of simple learning tasks considered in the paper allowed us to compare these results with the true learning curves (or their tight bounds) which can serve as benchmarks.

Figure 1.a implies that values of parameter $\omega$ in the $\omega$-uniformity bound (approximation) governing a distribution of error patterns between different error shells (c.f. [10]) has a

significant impact on learning curve shapes, changing from slow decrease to rapid jumps ("phase transitions") in generalization.

Figure 1.b proves that one loses tightness of the bound by using $|H|_i^m$ rather than $|H_\epsilon|_i^m$, and even more is lost if $\omega$-uniformity bounds (with variable $C_{\omega,m}$) are employed. Inspecting Figures 1.b and 2.a we find that approximate approaches consisting of replacing $|H_\epsilon|_i^m$ by a simple estimate ($\omega$-uniformity) can produce learning curves very close to $|H|_i^m$-learning curves suggesting that an application of this formalism to learning systems where neither $|H_\epsilon|_i^m$ nor $|H|_i^m$ can by calculated might be possible. This could lead to a sensible approximate theory capturing at least certain qualitative properties of learning curves for more complex learning tasks.

Generally, the results of this paper show that by incorporating the limited knowledge of the statistical distribution of error patterns in the sample space one can dramatically improve bounds on the learning curve with respect to the classical universal estimates of the VC-theory. This is particularly important for "practical" training sample sizes ($m \leq 12 \times$ VC-dimension) where the VC-bounds are void.

**Acknowledgement.** The permission of Director, Telstra Research Laboratories, to publish this paper is gratefully acknowledged. A.K. acknowledges the support of the Australian Research Council.

# References

[1] S. Amari, N. Fujita, and S. Shinomoto. Four types of learning curves. *Neural Computation*, 4(4):605–618, 1992.

[2] M. Anthony and N. Biggs. *Computational Learning Theory*. Cambridge University Press, 1992.

[3] A. Blumer, A. Ehrenfeucht, D. Haussler, and M.K. Warmuth. Learnability and the Vapnik-Chervonenkis dimensions. *Journal of the ACM*, **36**:929–965, (Oct. 1989).

[4] T.M. Cover. Geometrical and statistical properties of linear inequalities with applications to pattern recognition. *IEEE Trans. Elec. Comp.*, **EC-14**:326–334, 1965.

[5] D. Haussler, M. Kearns, H.S. Seung, and N. Tishby. Rigorous learning curve bounds from statistical mechanics. In *Proc. 7th Ann. ACM Conf. on Comp. Learn. Theory*, pages 76–87, 1994.

[6] A. Kowalczyk. Estimates of storage capacity of multi-layer perceptron with threshold logic hidden units. *Neural Networks*, to appear.

[7] A. Kowalczyk. VC-formalism with explicit bounds on error shells size distribution. A manuscript, 1994.

[8] A. Kowalczyk and H. Ferra. Generalisation in feedforward networks. *Adv. in NIPS 7, The MIT Press*, Cambridge, 1995.

[9] A. Kowalczyk, J. Szymanski, and H. Ferra. Combining statistical physics with VC-bounds on generalisation in learning systems. In *Proc. ACNN'95*, Sydney, 1995. University of Sydney.

[10] A. Kowalczyk, J. Szymanski, and R.C. Williamson. Learning curves from a modified vc-formalism: a case study. In *Proceedings of ICNN'95, Perth (CD-ROM)*, volume VI, pages 2939–2943, Rundle Mall, South Australia, 1995. IEEE/Causal Production.

[11] J.G. Mauldon. Random division of an interval. *Proc. Cambridge Phil. Soc.*, **47**:331–336, 1951.

[12] K.R. Muller, M. Finke, N. Murata, and S. Amari. On large scale simulations for learning curves. In *Proc. ACNN'95*, pages 45–48, Sydney, 1995. University of Sydney.

[13] A. Sakurai. n-h-1 networks store no less $n$ $h + 1$ examples but sometimes no more. In *Proceedings of the 1992 International Conference on Neural Networks*, pages III–936–III–941. IEEE, June 1992.

[14] H. Sompolinsky, H.S. Seung, and N. Tishby. Statistical mechanics of learning curves. *Physical Reviews*, **A45**:6056–6091, 1992.

[15] V. Vapnik. *Estimation of Dependences Based on Empirical Data*. Springer-Verlag, 1982.

[16] V. Vapnik, E. Levin, and Y. Le Cun. Measuring the VC-dimension of a learning machine. *Neural Computation*, **6** (5):851–876, 1994.

[17] C. Wang and S.S. Venkantesh. Temporal dynamics of generalisation in neural networks. *Adv. in NIPS 7, The MIT Press*, Cambridge, 1995.
